# Catastrophic Interference in Human Motor Learning

**Tom Brashers-Krug, Reza Shadmehr[†], and Emanuel Todorov**
Dept. of Brain and Cognitive Sciences, M. I. T., Cambridge, MA 02139
[†]Currently at Dept. of Biomedical Eng., Johns Hopkins Univ., Baltimore, MD 21205
Email: tbk@ai.mit.edu, reza@bme.jhu.edu, emo@ai.mit.edu

## Abstract

Biological sensorimotor systems are not static maps that transform input (sensory information) into output (motor behavior). Evidence from many lines of research suggests that their representations are plastic, experience-dependent entities. While this plasticity is essential for flexible behavior, it presents the nervous system with difficult organizational challenges. If the sensorimotor system adapts itself to perform well under one set of circumstances, will it then perform poorly when placed in an environment with different demands (negative transfer)? Will a later experience-dependent change undo the benefits of previous learning (catastrophic interference)? We explore the first question in a separate paper in this volume (Shadmehr et al. 1995). Here we present psychophysical and computational results that explore the question of catastrophic interference in the context of a dynamic motor learning task. Under some conditions, subjects show evidence of catastrophic interference. Under other conditions, however, subjects appear to be immune to its effects. These results suggest that motor learning can undergo a process of consolidation. Modular neural networks are well suited for the demands of learning multiple input/output mappings. By incorporating the notion of fast- and slow-changing connections into a modular architecture, we were able to account for the psychophysical results.

# 1    Introduction

Interacting physically with the world changes the dynamics of one's limbs. For example, when holding a heavy load, a different pattern of muscular activity is needed to move one's arm along a particular path than when not holding a load. Previous work in our laboratory has shown that humans learn a novel dynamic task by forming an internal model of the new inverse dynamics of thier limbs. (Shadmehr and Mussa-Ivaldi 1994, Shadmehr et al, 1995). Preliminary evidence suggests that subjects can retain one of these internal models over time (Brashers-Krug, et al. 1994). Humans are required, however, to move their limbs effectively under a large number of dynamic conditions. Are people able to learn and store an inverse dynamic model appropriate for each condition, or do they form such models from scratch as they need them? In particular, can learning a new inverse dynamic model overwrite or displace a previous model? We will present evidence that certain conditions must be met before a subject is able to retain more than one inverse dynamic model in a given experimental context. These conditions can be modeled as leading to a process of consolidation, whereby learning is transfered from vulnerable, low-capacity storage to a long-term, high-capacity storage.

# 2    Experimental Protocol

We have developed a motor learning paradigm that allows us to alter the dynamics of a subject's arm and so to monitor a subject's ability to learn dynamic tasks. A subject moves the handle on the free end of a two-link planar robot arm–called a manipulandum–to guide a cursor to a series of targets displayed on a computer screen (fig 1a). The position and velocity of the handle of the manipulandum are recorded at ten-millisecond intervals and are used to deliver state-dependent forces to the subject's hand. In order to test a subject's ability to learn a novel dynamic task, we present the subject with a viscous force field as s/he moves from one target to the next (fig 1b). Initially, such forces perturb the subject's movements, causing them to depart from the smooth, straight-line trajectories of the baseline condition (i.e., the condition before the viscous forces were presented) (figs 1c,1d). The extent of learning is measured as the degree to which a subject's movements in the force field over time come to resemble that subject's baseline movements. We have shown in previous work that subjects adapt to the imposed force fileds by forming a predictive model of the velocity-dependent forces, and that subjects use this inverse dynamic model to control their arms in what appears to be a feedforward manner (Shadmehr and Mussa-Ivaldi 1994).

# 3    Psychophysical Findings

## 3.1    Catastrophic Interference

Here, we employed this paradigm to explore the consequences of learning two different dynamic tasks in a row. In an initial series of experiments, we allowed twelve subjects to learn to move the manipulandum in a first force field (Field A) for approximately 5 minutes. Immediately after this first set of movements, we presented the subjects with an anti-correlated force field (Field B). For example, if we pre-

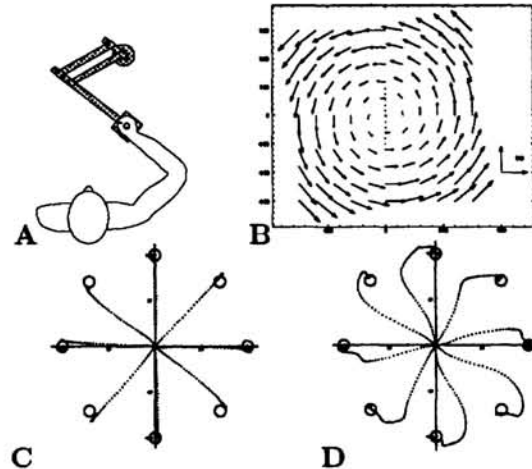

Figure 1: **A:** The experimental setup. **B:** An example of a viscous field, plotted in velocity space (mm/sec). The arrows indicate the direction and magnitude of the forces exerted by the manipulandum on the subject's hand at each location in velocity space. **C:** One subject's trajectories before forces were introduced. Targets are indicated by the open circles. **D:** Trajectories immediately after the force field in (B) was presented.

sented the counter-clockwise curl field depicted in fig. 1B as Field A, we would next present a clockwise curl field as Field B. Half the subjects learned the clockwise curl field first and the counter-clockwise field second; the other half learned the two fields in the reverse order. (The first field will be referred to as Field A and the second field as Field B, whichever field was learned first.) The subjects' mean performance in Field B was worse (p< 0.0001, paired t-test) than in Field A. This phenomenon has been called negative transfer in the psychophysical literature. Negative transfer in this motor learning paradigm is explored more fully in a separate paper in this volume (Shadmehr et al, 1995). In that paper, we suggested that this negative transfer could result from the fact that the same neural elements are learning both tasks. We predicted that, if this is the case, learning to move in Field B would interfere with a subject's ability to retain an inverse dynamic model of Field A. Learning to move in Field B would, in effect, cause subjects to "unlearn" Field A, resulting in catastrophic interference.

In order to test this prediction, we compared the improvement in performance from one day to the next of two groups of subjects, with twelve subjects in each group. The subjects in the control group learned to move in one force field on Day One and were then tested on Day Two in the same force field. The subjects in the experimental group learned two separate force fields in a row on Day One and were then tested on Day Two in the first force field they learned. The experimental group retained significantly less of their learning (p< 0.01, paired t-test) from Day One to Day Two than the control group (figs 2a,2b). In other words, learning the second force field resulted in catastrophic interference. (The question of whether this represents a storage or a retrieval phenomenon is beyond the scope of this paper.)

## 3.2   Consolidation

Having found evidence for catastrophic interference, we wanted to know whether there were circumstances under which dynamic motor learning was immune to being functionally erased by subsequent learning. We therefore tested two further groups of subjects. We allowed these subjects to practice longer in one field before they learned the second field. We also allowed 24 hours to pass between the time subjects first learned one field and when they learned the second field. The subjects in the experimental group (n = 10) practiced in one force field for approximately 15 minutes on Day One. They returned on Day Two and practiced in the same force field for five more minutes. They were then allowed to practice in a second force field for 15 minutes. By the end of this fifteen minutes, they were performing in the second field at a level comparable to the level they acheived in the first force field. We had the subjects return on Day Three, when they were tested for their retention of the first field they learned. We compared the difference in performance on Day Two and Day Three of this experimental group with that of a control group (n = 9) who followed the same protocol in all respects except that they never learned a second force field. In this way we could determine whether learning the second field resulted in a decrement in performance for the experimental group when compared with the control group.

Under these conditions, we found no difference in the retention of learning between the experimental and control groups (fig 2c, 2d). That is, learning the second field under these conditions no longer resulted in catastrophic interference. What subjects had learned about the first field had become resistant to such interference. It had become consolidated.

We can not tell from these experiments whether such consolidation is the result of the increased practice in the first field, or whether it is the result of the 24 hours that elapsed between when the first field was first learned and when the second field was learned. There is evidence that increased practice in a motor task can engage different neural circuits than those that are active during initial learning (Jenkins, et al 1994). The shift to "practiced" circuits may represent the neural substrate of consolidation. There is also evidence that time can be an important factor in the consolidation of skill learning. (Karni and Sagi 1993) In the next section, we present a model of our results that assumes that time is the key variable in the consolidation of motor learning. The model could also be applied to a practice-based model of consolidation with minor modifications.

## 4   Computational Modeling of the Experimental Results

In order to model the results presented above we need a network that learns to compute an appropriate control signal $Y$ given the current state and the desired next state $X$ of the plant. More precisely, it needs to compute a mapping from joint angles $\theta$, joint velocities $\dot{\theta}$, and desired joint accelerations $\ddot{\theta}$ to torques. Several approaches for solving this problem have been proposed. One way to learn a mapping from $X$ into $Y$ is to use direct inverse modeling: apply a control signal, measure the next state of the plant, and use the current state, new state, and control signal as a training pair for the controller. This approach is not suitable for explaining non-

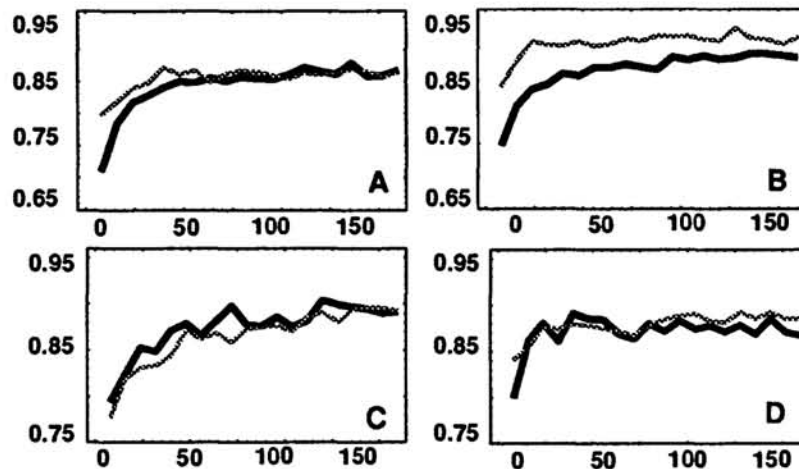

Figure 2: Plots of average learning curves. The correlation of trajectories in the force field to baseline trajectories (before the force field was applied) is plotted as a function of movement number. **A:** Learning curves for the first experimental group. Dark curve: learning curve on Day One in Field A. Light curve: learning curve in Field A on Day Two. This group learned Field B immediately after Field A on Day One (learning curve for Field B not shown). Note minimal improvement from Day One to Day Two. **B:** Learning curves for the first control group. Dark curve: learning curve in Field A on Day One. Light curve: learning curve in Field A on Day Two. Control group never learned Field B. Note significant improvement from Day One to Day Two. **C:** Second experimental group. Dark line: learning curve in Field A on Day Two immediately before learning Field B (Field B curve not shown). Light line: learning curve in Field A on Day Three. **D:** Second control group. Dark and light lines: learning curves in FIeld A on Day Two and Three, respectively. Note the similarity of the curves in C and D. This indicates that learning Field B did not significantly affect the experimental group's retention of Field A. All curves in C and D are significantly higher than curves for the initial learning of Field A on Day One.

convex mappings, however. The learning situation we must model is non-convex: we change the dynamic environment the controller operates in by presenting force fileds, and so there will be different $Y$ values corresponding to any $X$ value. A different approach that solves the nonconvexity problem is distal supervised learning (Jordan and Rumelhart 1992): produce control signals, observe the new state of the plant, and use that information to train a forward model that maps actions into states; then learn a controller, using the forward model to backpropagate error from observable cartesian coordinates to the unknown control space. Distal supervised learning solves the nonconvexity problem by learning one correct value of $Y$ for each $X$. But that can not explain the consolidation of learning – when the force field changes back to something already learned, our controller should rapidly recover its performance in that old field, meaning that it should retain information about all $Y$s that a particular $X$ can map into.

An architecture that seems to have most of the desirable properties discussed above is the Mixture of Experts (ME) model (Jordan and Jacobs 1994): several experts learn a mapping from $X$ to $Y$. A separate gating network selects an expert which is most likely to be correct at each moment. Such a model has been used previously (Jacobs and Jordan 1993) to learn a feedforward controller for a 2-joint planar arm operating under different loads. In their model however they assumed that the

identity of the load is known to the controller. The subjects in our study were not given any explicit cues about the identity of the fields they were learning. The mixture of experts cannot be used directly here because it decides which expert to select based on a soft partitioning of the input space, and in our experiment any force field is active over any portion of the input space at different moments in time. Here we propose an extension to the ME architecture that is able to deal with mappings overlapping in X space.

Another aspect of the results that is difficult to model using standard computational architectures is memory consolidation. To account for this effect we introduce two different learning rates (Alverez and Squire 1994). Some connections in the network change faster, as a result of which they can serve as short-term memory. We also introduce an off-line training phase (possibly corresponding to sleep) in which random inputs are generated, the part of the network containing the fast connections is used to produce a target output, and the resulting input-output pair is used to train the slowly changing connections. During the offline phase the faster changing connections are fixed, after that they are randomized.

### 4.1   Modified Mixture of Experts

The ME model assumes that $Y$ is generated from $X$ by one of $N$ processes $(W_1, ..., W_N)$ and therefore the likelihood function is:

$$L(\Theta|x_t, y_t) = P(y_t|x_t, \Theta) = \sum_i g_i{}^t P(y_t|W_i, x_t, \Theta)$$

$$g_i{}^t = P(W_i|x_t, \Theta),$$

where $\Theta$ represents the parameters of the model and $g_i$ is the prior probability. We want to use the posterior probability $P(W_i|x_t, y_t)$ instead, because the processes (different force fields) are separable in $XY$ space, but not in $X$ space. If we want to implement an on-line controller such a term is not available, because at time $t$, $y_t$ is still unknown (the task of the controller is to produce it). We could approximate $P(W_i|x_t, y_t)$ with $P(W_i|x_{t-1}, y_{t-1})$, because dynamic conditions do not change very often. Now the gating network (which computes $P(W_i|x_t)$ in ME) is going to select expert $i$ based on the previous $XY$ pair. This approach would obviously lead to a single large error at the moment when the force field changes, but so will any model using only $x_t$ to compute $y_t$. In fact such an error seems to be consistent with our psychophysics data. Thus the learning rule is:

$$\Delta\Theta_i = \nu_i h_i{}^t(y_t - \mu_i(x_t, \Theta_i))$$

$$g_i{}^t = h_i{}^{t-1}$$

$$h_i{}^t = P(W_i^t|x_i^t, y_i^t, \Theta) = \frac{g_i{}^t P(y_t|W_i, x_t, \Theta)}{\sum_j g_j{}^t P(y_t|W_j, x_t, \Theta)},$$

where $h_i^t$ is the posterior probability and $\mu_i$ is the output of exper $i$. $\mu_i$ is a linear function of the inputs. In our model we used 4 experts. In order to model the process of consolidation, we gave one expert a learning rate that was 10 times higher than the learning rate of the other 3 experts.

## 4.2   The Model

We simulated the dynamics of a 2-joint planar arm similar to the one used in our previous work (Shadmehr and Mussa-Ivaldi, 1994). The torque applied to the arm at every point in time is the sum of the outputs of a fixed controller, a PD controller, and an adaptive controller with the architecture described above.

The fixed controller maps $\theta, \dot{\theta}$ (current state), and $\ddot{\theta}$ (desired next state) into a torque $\tau$. The mapping is exact when no external forces are applied. The desired trajectories are minimum-jerk trajectories (Flash and Hogan 1985) sampled at 100Hz. The desired trajectories are 10 cm long and last 0.5 seconds. The PD controller is used to compensate for the troques produced by the force field while the adaptive controller has not yet learned to do that. The adaptive part of the controller consists of a mixture of 4 linear experts (whose initial output is small) and a modified gating network described above. The system operates as follows: $\theta, \dot{\theta}, \ddot{\theta}$ are sent to the fixed controller, which outputs a torque $\tau_1$; the PD controller outputs a torque $\tau_2$ based on the current deviation from the desired joint position and velocity; 8 terms describing the current state of the arm (and chosen to linearize the mapping to be learned) are sent to the mixture model, which outputs a torque $\tau_3$; $\tau_c = \tau_1 + \tau_2 - \tau_3$ is applied to the plant as a control signal; the actual torque $\tau = \tau_c + \tau_f$ is computed. The mixture model is trained to produce the torque $\tau_f$ resulting from the force field. In other words, the adaptive part of the controller learns to compensate for the force field exerted by the environment.

The parameters of the mixture model are updated after every movement, so a training pair $(x_t, y_t)$ is actually a batch of 50 points. The input, $x_t$, consists of the 8 terms describing the current state and the desired next state; the output, $y_t$, is the torque vector that the force field produces.The compensatory torques for a complete movement are computed before the movement starts. The only processing done during the movement is the computations necessary for the PD controller.

## 4.3   Results

### 4.3.1   Negative Transfer

When the network was given two successive, incompatable mappings to learn (this corresponds to learning to move in two opposite force fields), the resulting performance very much resembled that of our human subjects. The performance in the second mapping was much poorer than that in the first mapping. The fast-learning expert changed its weights to learn both tasks. Since the two tasks involved anti-correlated maps, the fast expert's weights after learning the first mapping were very inappropriate for the second task, leading to the observed negative transfer.

### 4.3.2   Catastrophic Interference

When the network was trained on two successive, opposite force fields, with no consolidation occurring between the two training sessions, the learning in second training session overwrote the learning that occurred during the first training session (fig 3A). Since the expert with the fast-changing weights attempted to learn both mappings, this catastrophic interference is not unexpected.

### 4.3.3   Consolidation

When the network was allowed to undergo consolidation between learning the first and the second force field, the network no longer suffered from catastrophic inter-

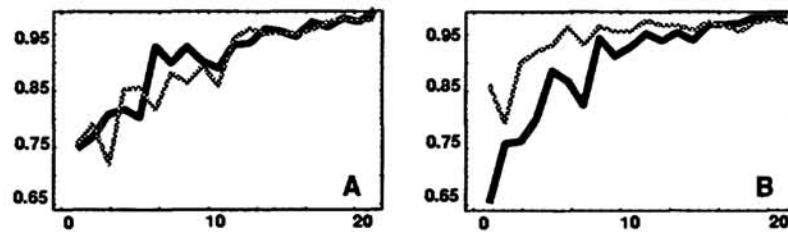

Figure 3: **A:** Learning curves for the ME architecture. Dark line: curve when first learning Field A. light line: curve when given Field A a second time, after learning FIeld B (no consolidation allowed between learning Field A and Field B). Note lack of retention of Field A. **B:** Learning curves for the same architecture in Field A before and after learning Field B. Consolidations was allowed between learning Field A and Field B.

ference (fig 3B). The learning that had initially resided in the fast-learning expert was transfered to one of the slower-learning networks. Thus, when the expert with the fast-changing connections learned the second mapping, the original learning was no longer destroyed. In addition, when the network was allowed to consolidate the second force field, a different slow-learning expert stored the second mapping. In this way, the network stored multiple maps in long-term memory.

## 5   Conclusions

We have presented psychophysical evidence for catastrophic interference. We have also shown results that suggest that motor learning can undergo a process of consolidation. By adding distinct fast- and slow-changing weights to a mixture of experts architecture, we were able to account for these psychophysical findings. We plan to investigate further the neural correlates of consolidation using both brain imaging in humans and electrophysiological studies in primates.

### References

P. Alverez and L. Squire. (1994) Memory consolidation and the medial temporal love: a simple network model, *PNAS***91:15**:7041-7045. T. Brashers-Krug, et al. (1994) Temporal aspects of human motor learning, *Soc. Neurosci. Abstract* in press.

T. Flash and N. Hogan. (1985) The coordination of arm movements: an experimentally confirmed mathematical model. *J. Neurosci.* **5**:1688-1703.

R. Jacobs and M. Jordan. (1993) Learning piecewise control strategies in a modular neural network architecture. IEEE Trans. on Systems, Man and Cyber. **23:2**: 337-345.

I. Jenkins, et al. (1994) Motor sequence learning: a study with positron emission tomography, *J. Neurosci.*14:3775-3790.

M. Jordan and R. Jacobs. (1994) Hierarchical mixture of experts and the EM algorithm, *Neural Computation* **6:2**: 181-214.

M. Jordan and D. Rumelhart. (1992) Forward models: supervised learning with a distal teacher *Cognitive Sci.* 16:307-354

A. Karni and D. Sagi. (1993) *Nature* **365**:250.

R. Shadmehr and F. Mussa-Ivaldi. (1994) Adaptive representation of dynamics during learning of a motor task, *J. Neurosci.*14:5:3208-3224.

R. Shadmehr, T. Brashers-Krug, F. Mussa-Ivaldi, (1995) Interference in learning internal models of inverse dynamics in humans, *Adv Neural Inform Proc Syst*vol 7, in press
